# Quadratic–Type Lyapunov Functions for Competitive Neural Networks with Different Time–Scales

**Anke Meyer–Bäse**
Institute of Technical Informatics
Technical University of Darmstadt
Darmstadt, Germany 64283

## Abstract

The dynamics of complex neural networks modelling the self–organization process in cortical maps must include the aspects of long and short–term memory. The behaviour of the network is such characterized by an equation of neural activity as a fast phenomenon and an equation of synaptic modification as a slow part of the neural system. We present a quadratic–type Lyapunov function for the flow of a competitive neural system with fast and slow dynamic variables. We also show the consequences of the stability analysis on the neural net parameters.

## 1 INTRODUCTION

This paper investigates a special class of laterally inhibited neural networks. In particular, we have examined the dynamics of a restricted class of laterally inhibited neural networks from a rigorous analytic standpoint.

The network models for retinotopic and somatotopic cortical maps are usually composed of several layers of neurons from sensory receptors to cortical units, with feedforward excitations between the layers and lateral (or recurrent) connection within the layer. Standard techniques include (1) Hebbian rule and its variations for modifying synaptic efficacies, (2) lateral inhibition for establishing topographical organization of the cortex, and (3) adiabatic approximation in decoupling the dynamics of relaxation (which is on the fast time scale) and the dynamics of learning (which is on the slow time scale) of the network. However, in most cases, only computer simulation results were obtained and therefore provided limited mathematical understanding of the self–organizing neural response fields.

The networks under study model the dynamics of both the neural activity levels,

the short–term memory (STM), and the dynamics of synaptic modifications, the long–term memory (LTM). The actual network models under consideration may be considered extensions of Grossberg's shunting network [Gro76] or Amari's model for primitive neuronal competition [Ama82]. These earlier networks are considered pools of mutually inhibitory neurons with **fixed** synaptic connections. Our results extended these earlier studies to systems where the synapses can be modified by external stimuli. The dynamics of competitive systems may be extremely complex, exhibiting convergence to point attractors and periodic attractors. For networks which model only the dynamic of the neural activity levels Cohen and Grossberg [CG83] found a Lyapunov function as a necessary condition for the convergence behavior to point attractors.

In this paper we apply the results of the theory of Lyapunov functions for singularly perturbed systems on large–scale neural networks, which have two types of state variables (LTM and STM) describing the slow and the fast dynamics of the system. So we can find a Lyapunov function for the neural system with different time–scales and give a design concept of storing desired pattern as stable equilibrium points.

## 2   THE CLASS OF NEURAL NETWORKS WITH DIFFERENT TIME–SCALES

This section defines the network of differential equations characterizing laterally inhibited neural networks. We consider a laterally inhibited network with a deterministic signal Hebbian learning law [Heb49] and is similar to the spatiotemporal system of Amari [Ama83].

The general neural network equations describe the temporal evolution of the STM (activity modification) and LTM states (synaptic modification). For the $j$th neuron of a $N$–neuron network these equations are:

$$\dot{x}_j = -a_j x_j + \sum_{i=1}^{N} D_{ij} f(x_i) + B_j S_j \qquad (1)$$

$$\dot{S}_j = -S_j + |\mathbf{y}|^2 f(x_j) \qquad (2)$$

where $x_j$ is the current activity level, $a_j$ is the time constant of the neuron, $B_j$ is the contribution of the external stimulus term, $f(x_i)$ is the neuron's output, $D_{ij}$ is the lateral inhibition term and $y_i$ is the external stimulus. The dynamic variable $S_j$ represents the synaptic modification state and $|\mathbf{y}|^2|$ is defined as $|\mathbf{y}|^2 = \mathbf{y}^T \mathbf{y}$.

We will assume that the input stimuli are normalized vectors of unit magnitude $|\mathbf{y}|^2 = 1$. These systems will be subject to our analysis considerations regarding the stability of their equilibrium points.

## 3   ASYMPTOTIC STABILITY OF NEURAL NETWORKS WITH DIFFERENT TIME–SCALES

We show in this section that it is possible to determine the asymptotic stability of this class of neural networks interpreting them as nonlinear singularly perturbed systems. While singular perturbation theory, a traditional tool of fluid dynamics and nonlinear mechanics, embraces a wide variety of dynamic phenomena possesing slow and fast modes, we show that singular perturbations are present in many

neurodynamical problems. In this sense we apply in this paper the results of this valuable analysis tool on the dynamics of laterally inhibited networks.

In [SK84] is shown that a quadratic–type Lyapunov function for a singularly perturbed system is obtained as a weighted sum of quadratic–type Lyapunov functions of two lower order systems: the so–called reduced and the boundary–layer systems. Assuming that each of the two systems is asymptotically stable and has a Lyapunov function, conditions are derived to guarantee that, for a sufficiently small perturbation parameter, asymptotic stability of the singularly perturbed system can be established by means of a Lyapunov function which is composed as a weighted sum of the Lyapunov functions of the reduced and boundary–layer systems.

Adopting the notations from [SK84] we will consider the singularly perturbed system [2]

$$\dot{\mathbf{x}} = \mathbf{f}(\mathbf{x}, \mathbf{y}) \quad \mathbf{x} \in \mathbf{B_x} \subset \mathbf{R^n} \tag{3}$$

$$\epsilon \dot{\mathbf{y}} = \mathbf{g}(\mathbf{x}, \mathbf{y}, \epsilon) \quad \mathbf{y} \in \mathbf{B_y} \subset \mathbf{R^m}, \epsilon > 0 \tag{4}$$

We assume that, in $\mathbf{B_x}$ and $\mathbf{B_y}$, the origin ($\mathbf{x} = \mathbf{y} = \mathbf{0}$) is the unique equilibrium point and (3) and (4) has a unique solution. A reduced system is defined by setting $\epsilon = 0$ in (3) and (4) to obtain

$$\dot{\mathbf{x}} = \mathbf{f}(\mathbf{x}, \mathbf{y}) \tag{5}$$

$$\mathbf{0} = \mathbf{g}(\mathbf{x}, \mathbf{y}, 0) \tag{6}$$

Assuming that in $\mathbf{B_x}$ and $\mathbf{B_y}$, (6) has a unique root $\mathbf{y} = \mathbf{h}(\mathbf{x})$, the reduced system is rewritten as

$$\dot{\mathbf{x}} = \mathbf{f}(\mathbf{x}, \mathbf{h}(\mathbf{x})) = \mathbf{f_r}(\mathbf{x}) \tag{7}$$

A boundary–layer system is defined as

$$\frac{\partial \mathbf{y}}{\partial \tau} = \mathbf{g}(\mathbf{x}, \mathbf{y}(\tau), 0) \tag{8}$$

where $\tau = t/\epsilon$ is a stretching time scale. In (8) the vector $\mathbf{x} \in \mathbf{R^n}$ is treated as a fixed unknown parameter that takes values in $\mathbf{B_x}$. The aim is to establish the stability properties of the singularly perturbed system (3) and (4), for small $\epsilon$, from those of the reduced system (7) and the boundary–layer system (8). The Lyapunov functions for system 7 and 8 are of quadratic–type. In [SK84] it is shown that under mild assumptions, for sufficiently small $\epsilon$, any weighted sum of the Lyapunov functions of the reduced and boundary–layer system is a quadratic-type Lyapunov function for the singularly perturbed system (3) and (4).

The necessary assumptions are stated now [SK84]:

1. The reduced system (7) has a Lyapunov function $V : \mathbf{R^n} \to \mathbf{R_+}$ such that for all $\mathbf{x} \in \mathbf{B_x}$

$$(\nabla_x V(\mathbf{x}))^T \mathbf{f_r}(\mathbf{x}) \leq -\alpha_1 \psi^2(\mathbf{x}) \quad \alpha_1 > 0 \tag{9}$$

where $\psi(\mathbf{x})$ is a scalar-valued function of $\mathbf{x}$ that vanishes at $\mathbf{x} = \mathbf{0}$ and is different from zero for all other $\mathbf{x} \in \mathbf{B_x}$. This condition guarantees that $\mathbf{x} = \mathbf{0}$ is an asymptotically stable equilibrium point of the reduced system (7).

2. The boundary–layer system (8) has a Lyapunov function $W(\mathbf{x}, \mathbf{y}) : \mathbf{R}^n \times \mathbf{R}^m \to \mathbf{R}_+$ such that for all $\mathbf{x} \in \mathbf{B_x}$ and $\mathbf{y} \in \mathbf{B_y}$

$$(\nabla_y W(\mathbf{x}, \mathbf{y}))^T \mathbf{g}(\mathbf{x}, \mathbf{y}, \mathbf{0}) \leq -\alpha_2 \phi^2(\mathbf{y} - \mathbf{h}(\mathbf{x})) \quad \alpha_2 > 0 \tag{10}$$

where $\phi(\mathbf{y} - \mathbf{h}(\mathbf{x}))$ is a scalar–valued function $(\mathbf{y} - \mathbf{h}(\mathbf{x})) \in \mathbf{R}^m$ that vanishes at $\mathbf{y} = \mathbf{h}(\mathbf{x})$ and is different from zero for all other $\mathbf{x} \in \mathbf{B_x}$ and $\mathbf{y} \in \mathbf{B_y}$. This condition guarantees that $\mathbf{y} = \mathbf{h}(\mathbf{x})$ is an asymptotically stable equilibrium point of the boundary–layer system (8).

3. The following three inequalities hold $\forall \mathbf{x} \in \mathbf{B_x}$ and $\forall \mathbf{y} \in \mathbf{B_y}$:

   a.)

$$(\nabla_x W(\mathbf{x}, \mathbf{y}))^T \mathbf{f}(\mathbf{x}, \mathbf{y}) \leq c_1 \phi^2(\mathbf{y} - \mathbf{h}(\mathbf{x})) + c_2 \psi(\mathbf{x}) \phi(\mathbf{y} - \mathbf{h}(\mathbf{x})) \tag{11}$$

   b.)

$$(\nabla_x V(\mathbf{x}))^T [\mathbf{f}(\mathbf{x}, \mathbf{y}) - \mathbf{f}(\mathbf{x}, \mathbf{h}(\mathbf{x}))] \leq \beta_1 \psi(\mathbf{x}) \phi(\mathbf{y} - \mathbf{h}(\mathbf{x})) \tag{12}$$

   c.)

$$(\nabla_y W(\mathbf{x}, \mathbf{y}))^T [\mathbf{g}(\mathbf{x}, \mathbf{y}, \epsilon) - \mathbf{g}(\mathbf{x}, \mathbf{y}, \mathbf{0})] \leq$$
$$\epsilon K_1 \phi^2(\mathbf{y} - \mathbf{h}(\mathbf{x})) \quad + \quad \epsilon K_2 \psi(\mathbf{x}) \phi(\mathbf{y} - \mathbf{h}(\mathbf{x})) \tag{13}$$

The constants $c_1, c_2, \beta_1, K_1$ and $K_2$ are nonnegative. The inequalities above determine the permissible interaction between the slow and fast variables. They are basically smoothness requirements of $f$ and $g$.

After these introductory remarks the stability criterion is now stated:

**Theorem:** *Suppose that conditions 1–3 hold; let $d$ be a positive number such that $0 < d < 1$, and let $\epsilon^*(d)$ be the positive number given by*

$$\epsilon^*(d) = \frac{\alpha_1 \alpha_2}{\alpha_1 \gamma + [\beta_1(1 - d) + \beta_2 d]^2 / 4d(1 - d)} \tag{14}$$

*where $\beta_2 = K_2 + C_2$, $\gamma = K_1 + C_1$, then for all $\epsilon < \epsilon^*(d)$, the origin $(\mathbf{x} = \mathbf{y} = \mathbf{0})$ is an asymptotically stable equilibrium point of (3) and (4) and*

$$v(\mathbf{x}, \mathbf{y}) = (1 - d)V(\mathbf{x}) + dW(\mathbf{x}, \mathbf{y}) \tag{15}$$

*is a Lyapunov function of (3) and (4).*

If we put $\epsilon = \frac{1}{A}$ as a global neural time constant in equation (1) then we have to determine two Lyapunov functions: one for the boundary–layer system and the other for the reduced–order system.

In [CG83] is mentioned a global Lyapunov function for a competitive neural network with only an activation dynamics.

$$V(x) = -\sum_{i=1}^{n} \int_0^{x_i} b_i(\zeta_i) f_i'(\zeta_i) d\zeta_i + \frac{1}{2} \sum_{j,k=1}^{n} m_{jk} f_j(x_j) f_k(x_k) \tag{16}$$

under the constraints: $m_{ij} = m_{ji}$, $a_i(x_i) \geq 0$, $f_j(x_j) \geq 0$.

This Lyapunov–function can be taken as one for the boundary–layer system (STM–equation) , if the LTM contribution $S_i$ is considered as a fixed unknown parameter:

$$W(\mathbf{x}, \mathbf{S}) = \sum_{j=1}^{N} \int_{0}^{x_j} a_j(\zeta_j) f_j'(\zeta_j) d\zeta_j - \sum_{j=1}^{N} B_j S_j \int_{0}^{x_j} f_j'(\zeta_j) d\zeta_j - \frac{1}{2} \sum_{j=1}^{N} D_{ij} f_j(x_j) f_k(x_k)$$

(17)

For the reduced–order system (LTM–equation) we can take as a Lyapunov–function:

$$V(\mathbf{S}) = \frac{1}{2}\mathbf{S}^T\mathbf{S} = \sum_{i=1}^{N} S_i^2$$

(18)

The Lyapunov–function for the coupled STM and LTM dynamics is the sum of the two Lyapunov–function:

$$v(\mathbf{x}, \mathbf{S}) = (1 - d)V(\mathbf{S}) + dW(\mathbf{x}, \mathbf{S})$$

(19)

## 4  DESIGN OF STABLE COMPETITIVE NEURAL NETWORKS

Competitive neural networks with learning rules have moving equilibria during the learning process. The concept of asymptotic stability derived from matrix perturbation theory can capture this phenomenon.

We design in this section a competitive neural network that is able to store a desired pattern as a stable equilibrium.

The theoretical implications are illustrated in an example of a two neuron network.

*Example:* Let $N = 2$, $a_i = A$, $B_j = B$, $D_{ii} = \alpha > 0$, $D_{ij} = -\beta < 0$ and the nonlinearity be a linear function $f(x_j) = x_j$ in equations (1) and (2).

We get for the boundary–layer system:

$$\dot{x}_j = -Ax_j + \sum_{i=1}^{N} D_{ij} f(x_i) + BS_j$$

(20)

and for the reduced–order system:

$$\dot{S}_j = S_j[\frac{B}{A - \alpha} - 1] - \frac{C}{A - \alpha}$$

(21)

Then we get for the Lyapunov–functions:

$$V(\mathbf{S}) = \frac{1}{2}(S_1^2 + S_2^2)$$

(22)

and

$$W(\mathbf{x}, \mathbf{S}) = \frac{A}{2}x_1^2 + \frac{A}{2}x_2^2 - BS_1x_1 - BS_2x_2 - \frac{1}{2}[\alpha x_1^2 + \alpha x_2^2 - 2\beta x_1 x_2]$$

(23)

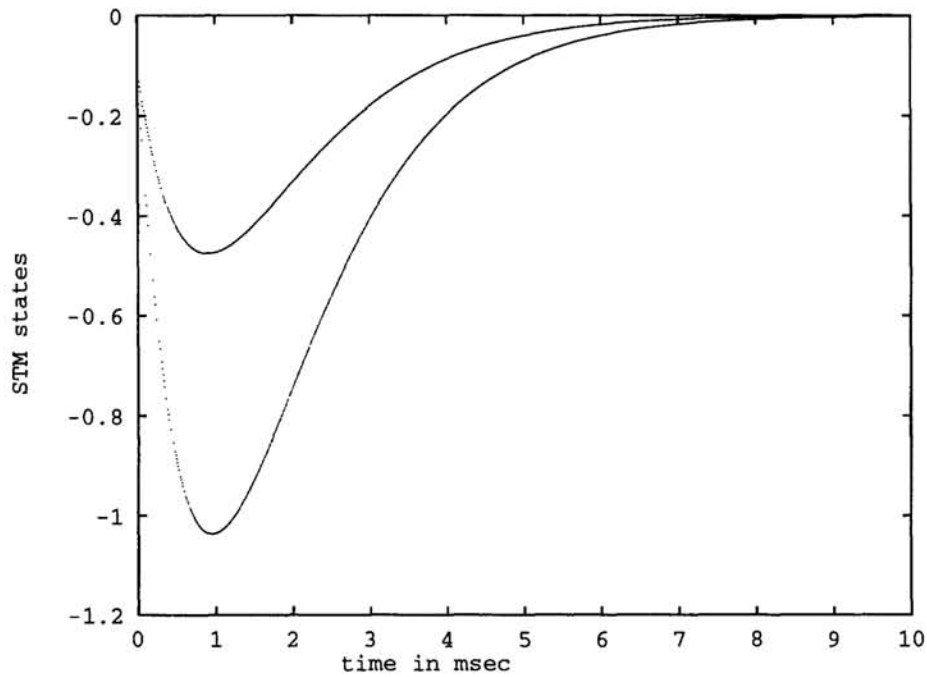

Figure 1: Time histories of the neural network with the origin as an equilibrium point: STM states.

For the nonnegative constants we get: $\alpha_1 = 1 - \frac{B}{A-\alpha}$, $\alpha_2 = (A-\alpha)^2$, $c_1 = \gamma = -B$, with $B < 0$, and $c_2 = \beta_1 = \beta_2 = 1$ and $K_1 = K_2 = 0$.

We get some interesting implications from the above results as: $A - \alpha > B$, $A - \alpha > 0$ and $B < 0$.

The above impications can be interpreted as follows: To achieve a stable equilibrium point $(0, 0)$ we should have a negative contribution of the external stimulus term and the sum of the excitatory and inhibitory contribution of the neurons should be less than the time constant of a neuron. An evolution of the trajectories of the STM and LTM states for a two neuron system is shown in figure 1 and 2. The STM states exhibit first an oscillation from the expected equilibrium point, while the LTM states reach monotonically the equilibrium point. We can see from the pictures that the equilibrium point $(0, 0)$ is reached after 5 msec by the STM- and LTM–states.

Choosing $B = -5$, $A = 1$ and $\alpha = 0.5$ we obtain for $\epsilon^*(d)$: $\epsilon^*(d) = \frac{11 \cdot 0.5^2}{55 + \frac{1}{4d(1-d)}}$.

From the above formula we can see that $\epsilon^*(d)$ has a maximum at $d = d^* = 0.5$.

## 5   CONCLUSIONS

We presented in this paper a quadratic–type Lyapunov function for analyzing the stability of equilibrium points of competitive neural networks with fast and slow dynamics. This global stability analysis method is interpreting neural networks as nonlinear singularly perturbed systems. The equilibrium point is constrained to a neighborhood of $(0, 0)$. This technique supposes a monotonically increasing non–linearity and a symmetric lateral inhibition matrix. The learning rule is a deterministic Hebbian. This method gives an upper bound on the perturbation

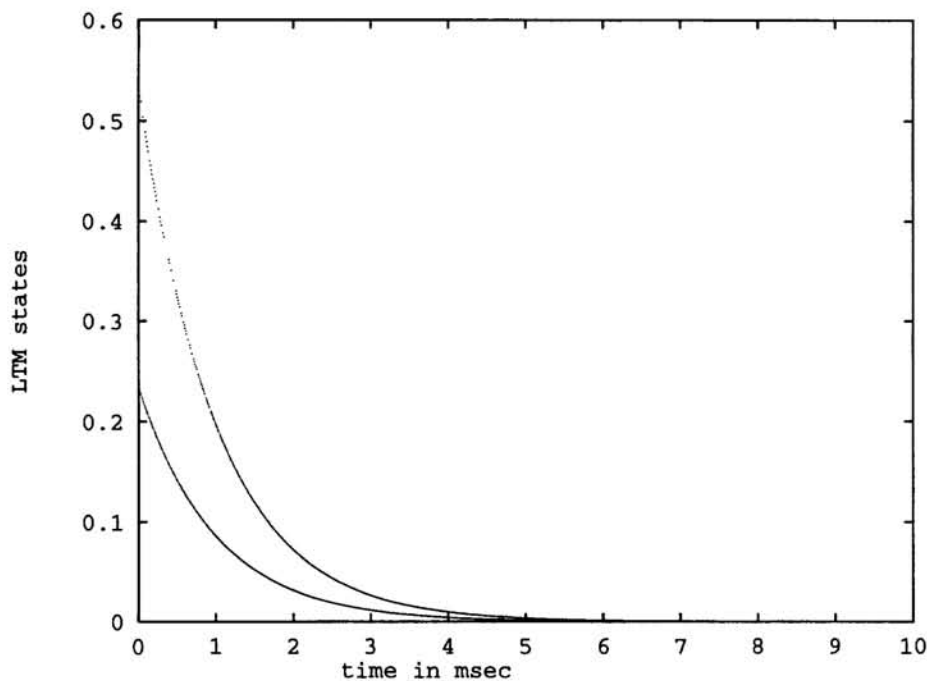

Figure 2: Time histories of the neural network with the origin as an equilibrium point: LTM states.

parameter and such an estimation of a maximal positive neural time–constant. The practical implication of the theoretical problem is the design of a competitive neural network that is able to store a desired pattern as a stable equilibrium.

## Footnotes

[2]The symbol $\mathbf{B_x}$ indicates a closed sphere centered at $\mathbf{x} = \mathbf{0}$; $\mathbf{B_y}$ is defined in the same way.

# References

[Ama82] S. Amari. *Competitive and cooperative aspects in dynamics of neural excitation and self–organization. Competition and cooperation in neural networks*, 20:1–28, 7 1982.

[Ama83] S. Amari. *Field theory of self–organizing neural nets. IEEE Transactions on systems, machines and communication*, SMC-13:741–748, 7 1983.

[CG83] A. M. Cohen und S. Grossberg. *Absolute Stability of Global Pattern Formation and Parallel Memory Storage by Competitive Neural Networks. IEEE Transactions on Systems, Man and Cybernetics*, SMC-13:815–826, 9 1983.

[Gro76] S. Grossberg. *Adaptive Pattern Classification and Universal Recording. Biological Cybernetics*, 23:121–134, 1 1976.

[Heb49] D. O. Hebb. *The Organization of Behavior*. J. Wiley Verlag, 1949.

[SK84] Ali Saberi und Hassan Khalil. *Quadratic–Type Lyapunov Functions for Singularly Perturbed Systems. IEEE Transactions on Automatic Control*, pp. 542–550, June 1984.
